# The Generalisation Cost of RAMnets

**Richard Rohwer and Michał Morciniec**
rohwerrj@cs.aston.ac.uk   morcinim@cs.aston.ac.uk
Neural Computing Research Group
Aston University
Aston Triangle, Birmingham B4 7ET, UK.

## Abstract

Given unlimited computational resources, it is best to use a criterion of minimal expected generalisation error to select a model and determine its parameters. However, it may be worthwhile to sacrifice some generalisation performance for higher learning speed. A method for quantifying sub-optimality is set out here, so that this choice can be made intelligently. Furthermore, the method is applicable to a broad class of models, including the ultra-fast memory-based methods such as RAMnets. This brings the added benefit of providing, for the first time, the means to analyse the generalisation properties of such models in a Bayesian framework.

## 1   Introduction

In order to quantitatively predict the performance of methods such as the ultra-fast RAMnet, which are not trained by minimising a cost function, we develop a Bayesian formalism for estimating the generalisation cost of a wide class of algorithms.

We consider the noisy interpolation problem, in which each output data point $y^i$ results from adding noise to the result $y = f(x)$ of applying unknown function $f$ to input data point $x$, which is generated from a distribution $P(x)$. We follow a similar approach to (Zhu & Rohwer, to appear 1996) in using a Gaussian process to define a prior over the space of functions, so that the expected generalisation cost under the posterior can be determined. The optimal model is defined in terms of the restriction of this posterior to the subspace defined by the model. The optimum is easily determined for linear models over a set of basis functions. We go on to compute the generalisation cost (with an error bar) for all models of this class, which we demonstrate to include the RAMnets.

Section 2 gives a brief overview of RAMnets. Sections 3 and 4 supply the formalism for computing expected generalisation costs under Gaussian process priors. Numerical experiments with this formalism are presented in Section 5. Finally, we discuss the current limitations of this technique and future research directions in Section 6.

## 2    RAMnets

The RAMnet, or n-tuple network is a very fast 1-pass learning system that often gives excellent results competitive with slower methods such as Radial Basis Function networks or Multi-layer Perceptrons (Rohwer & Morciniec, 1996). Although a semi-quantitative theory explains how these systems generalise, no formal framework has previously been given to precisely predict the accuracy of n-tuple networks.

Essentially, a RAMnet defines a set of "features" which can be regarded as Boolean functions of the input variables. Let the $a^{\underline{th}}$ feature of $x$ be given by a $\{0,1\}$-valued function $\phi_a(x)$. We will focus on the n-tuple regression network (Allinson & Kołcz, 1995), which outputs

$$y = \overset{m}{\mathbf{f}}(x) = \frac{\sum_a \phi_a(x) \sum_i y^i \phi_a(x^i)}{\sum_a \phi_a(x) \sum_i \phi_a(x^i)} = \frac{\sum_i y^i U(x, x^i)}{\sum_i U(x, x^i)} \tag{1}$$

in response to input $x$, if trained on the set of $N$ samples $\{\mathbf{x}_{(N)}\mathbf{y}_{(N)}\} = \{(x^i, y^i)\}_{i=1}^N$. Here $U(x, x') = \sum_a \phi_a(x)\phi_a(x')$ can be seen to play the role of a smoothing kernel, provided that it turns out to have a suitable shape. It is well-know that it does, for appropriate choices of feature sets. The strength of this method is that the sums over training data can be done in one pass, producing a table containing two totals for each feature. Only this table is required for recognition.

It is interesting to note that there is a familiar way to expand a kernel into the form $U(x, x') = \sum_a \phi_a(x)\phi_a(x')$, at least when $U(x, x') = U(x - x')$, if the range of $\phi$ is not restricted to $\{0, 1\}$: an eigenfunction expansion[1]. Indeed, principal component analysis[2] applied to a Gaussian with variance $\mathbf{V}$ shows that the smallest feature set for a given generalisation cost consists of the (real-valued) projections onto the leading eigenfunctions of $\mathbf{V}$. Be that as it may, the treatment here applies to arbitrary feature sets.

## 3    Bayesian inference with Gaussian priors

Gaussian processes provide a diverse set of priors over function spaces. To avoid mathematical details of peripheral interest, let us approximate the infinite-dimensional space of functions by a finite-dimensional space of discretised functions, so that function $f$ is replaced by high-dimensional vector $\mathbf{f}$, and $f(x)$ is replaced by $\mathbf{f}_x$, with $f(x) \approx \mathbf{f}_x$ within a volume $\Delta x$ around $x$. We develop the case of scalar functions $f$, but the generalisation to vector-valued functions is straightforward.

We assume a Gaussian prior on $\mathbf{f}$, with zero mean and covariance $\mathbf{V}/\alpha$:

$$P\left(\mathbf{f}\right) = (1/Z_\alpha)e^{-\frac{\alpha}{2}\mathbf{f}^T\mathbf{V}^{-1}\mathbf{f}} \qquad (2)$$

where $Z_\alpha = \det(\frac{2\pi}{\alpha}\mathbf{V})^{\frac{1}{2}}$. The overall scale of variation of $\mathbf{f}$ is controlled by $\alpha$. Illustrative samples of the functions generated from various choices of covariance are given in (Zhu & Rohwer, to appear 1996). With $q_x/\beta$ denoting the (possibly position-dependent) variance of the Gaussian output noise, the likelihood of outputs $\mathbf{y}_{(N)}$ given function $\mathbf{f}$ and inputs $\mathbf{x}_{(N)}$ is

$$P\left(\mathbf{y}_{(N)}|\mathbf{x}_{(N)},\mathbf{f}\right) = (1/Z_\beta)\exp-\frac{\beta}{2}\sum_i(\mathbf{f}_{x^i} - y^i)q_{x^i}^{-1}(\mathbf{f}_{x^i} - y^i) \qquad (3)$$

where $Z_\beta^2 = \prod_i \frac{2\pi}{\beta}q_{x^i} = \det\left[\frac{2\pi}{\beta}\mathbf{Q}\right]$ with $\mathbf{Q}_{ij} = q_{x^i}\delta_{ij}$.

Because $\mathbf{f}$ and $\mathbf{x}_{(N)}$ are independent the joint distribution is

$$P\left(\mathbf{y}_{(N)},\mathbf{f}|\mathbf{x}_{(N)}\right) = P\left(\mathbf{y}_{(N)}|\mathbf{f},\mathbf{x}_{(N)}\right)P\left(\mathbf{f}\right) = (e^{\frac{1}{2}\mathbf{b}^T\mathbf{A}\mathbf{b}+c})/(Z_\alpha Z_\beta)e^{-\frac{1}{2}(\mathbf{f}-\mathbf{A}\mathbf{b})^T\mathbf{A}^{-1}(\mathbf{f}-\mathbf{A}\mathbf{b})} \qquad (4)$$

where $\delta_{x,x^i}$ is understood to be 1 whenever $x^i$ is in the same cell of the discretisation as $x$, and $\mathbf{A}_{xx'}^{-1} = \alpha\mathbf{V}_{xx'}^{-1} + \beta\sum_i q_{x^i}^{-1}\delta_{x,x^i}\delta_{x',x^i}$, $\mathbf{b}_x = \beta\sum_i y^i q_{x^i}^{-1}\delta_{x,x^i}$, and $c = -\frac{1}{2}\beta\sum_i y^i q_{x^i}^{-1}y^i$. One can readily verify that

$$\mathbf{A}_{xx'} = (1/\alpha)\mathbf{V}_{xx'} + \sum_{tu}\mathbf{V}_{xx^t}\mathbf{K}_{tu}\mathbf{V}_{x^u x'} \qquad (5)$$

where $\mathbf{K}$ is the $N \times N$ matrix defined by

$$\mathbf{K}_{tu}^{-1} = -(\alpha^2/\beta)q_{x^t}\delta_{t,u} - \alpha\mathbf{V}_{x^t x^u}. \qquad (6)$$

The posterior is readily determined to be

$$P\left(\mathbf{f}|\mathbf{x}_{(N)},\mathbf{y}_{(N)}\right) = \frac{P\left(\mathbf{y}_{(N)},\mathbf{f}|\mathbf{x}_{(N)}\right)}{P\left(\mathbf{y}_{(N)}|\mathbf{x}_{(N)}\right)} = \det\left[2\pi\mathbf{A}\right]^{-\frac{1}{2}}e^{-\frac{1}{2}(\mathbf{f}-\overset{\bullet}{\mathbf{f}})^T\mathbf{A}^{-1}(\mathbf{f}-\overset{\bullet}{\mathbf{f}})}. \qquad (7)$$

where $\overset{\bullet}{\mathbf{f}} = \mathbf{A}\mathbf{b}$ is the posterior mean estimate of the true function $\mathbf{f}$.

## 4 Calculation of the expected cost and its variance

Let us define the cost of associating an output $\overset{m}{\mathbf{f}}_x$ of the model with an input $x$ that actually produced an output $y$ as

$$C(\overset{m}{\mathbf{f}}_x, y) = \frac{1}{2}(\overset{m}{\mathbf{f}}_x - y)^2 r_x$$

where $r_x$ is a position dependent cost weight.

The average of this cost defines a cost functional, given input data $\mathbf{x}_{(N)}$:

$$C(\overset{m}{\mathbf{f}}, \mathbf{f}|\mathbf{x}_{(N)}) = \int C(\overset{m}{\mathbf{f}_x}, y) P\left(x|\mathbf{x}_{(N)}\right) P\left(y|x, \mathbf{f}\right) dx dy. \tag{8}$$

This form is obtained by noting that the function $\mathbf{f}$ carries no information about the input point $x$, and the input data $\mathbf{x}_{(N)}$ supplies no information about $y$ beyond that supplied by $\mathbf{f}$. The distributions in (8) are unchanged by further conditioning on $\mathbf{y}_{(N)}$, so we could write $C(\overset{m}{\mathbf{f}}, \mathbf{f}|\mathbf{x}_{(N)}) = C(\overset{m}{\mathbf{f}}, \mathbf{f}|\mathbf{x}_{(N)}, \mathbf{y}_{(N)})$. This cost functional therefore has the posterior expectation value

$$\langle C|\mathbf{x}_{(N)}, \mathbf{y}_{(N)} \rangle = \int C(\overset{m}{\mathbf{f}}_x, y|\mathbf{x}_{(N)}) P\left(x, y, \mathbf{f}|\mathbf{x}_{(N)}, \mathbf{y}_{(N)}\right) dx dy d\mathbf{f} \tag{9}$$

and variance

$$\mathrm{var}\left[C(\overset{m}{\mathbf{f}}, \mathbf{f}|\mathbf{x}_{(N)})\right] = \int C(\overset{m}{\mathbf{f}}, \mathbf{f}|\mathbf{x}_{(N)})^2 P\left(\mathbf{f}|\mathbf{x}_{(N)}, \mathbf{y}_{(N)}\right) d\mathbf{f} - \langle C|\mathbf{x}_{(N)}, \mathbf{y}_{(N)} \rangle^2. \tag{10}$$

Plugging in the distributions (2) (applied to a single sample), (3) and (7) leads to:

$$\langle C|\mathbf{x}_{(N)}, \mathbf{y}_{(N)} \rangle = \tfrac{1}{2}\mathrm{tr}\left[\mathbf{AR}\right] + \tfrac{1}{2}(\overset{*}{\mathbf{f}} - \overset{m}{\mathbf{f}})^{\mathrm{T}}\mathbf{R}(\overset{*}{\mathbf{f}} - \overset{m}{\mathbf{f}}) + \frac{\mathrm{tr}\left[\mathbf{QR}\right]}{2\beta} \tag{11}$$

where the diagonal matrices $\mathbf{R}$ and $\mathbf{Q}$ have the elements $\mathbf{R}_{xx'} = P(x|X)r_x\Delta x\delta_{x,x'}$ and $\mathbf{Q}_{xx'} = q_x\delta_{xx'}$.

Similar calculations lead to the expression for the variance

$$\mathrm{var}\left[C(\overset{m}{\mathbf{f}}, \mathbf{f}|\mathbf{x}_{(N)}, \mathbf{y}_{(N)})\right] = \tfrac{1}{2}\mathrm{tr}\left[\mathbf{ARAR}\right] + \mathrm{tr}\left[\mathbf{ARRFF}\right]. \tag{12}$$

where the elements of $\mathbf{F}$ are $\mathbf{F}_{xx'} = (\overset{m}{\mathbf{f}_x} - \overset{*}{\mathbf{f}_x})\delta_{x,x'}$.

Note that the RAMnet (1) has the form $\overset{m}{\mathbf{f}_x} = \sum_i \mathbf{J}_{xx^i}y^i$ linear in the output data $\mathbf{y}_{(N)}$, with $\mathbf{J}_{xx^i} = U(x, x^i)/\sum_j U(x, x_j)$. Let us take $\mathbf{V}$ to have the form $\mathbf{V}(x, x') = p(x)G(x - x')p(x')$, combining translation-invariant and non-invariant factors in a plausible way. Then with the sums over $x$ replaced by integrals, (11) becomes explicitly

$$\begin{aligned}
2\langle C|\mathbf{x}_{(N)}, \mathbf{y}_{(N)} \rangle &= \frac{1}{\beta}\int dx P\left(x|\mathbf{x}_{(N)}\right) q_x r_x + \frac{1}{\alpha}\int P\left(x|\mathbf{x}_{(N)}\right) r_x p_x^2 \mathbf{G}_{xx} \\
&+ \frac{1}{\alpha}\sum_{tu} p_{x^t}\mathbf{K}_{tu}p_{x^u}\int dx P\left(x|\mathbf{x}_{(N)}\right) r_x p_x^2 \mathbf{G}_{x^u x}\mathbf{G}_{xx^t} \\
&+ \alpha^2 \sum_{tuvs} y^u \mathbf{K}_{ut}p_{x^t}\int dx P\left(x|\mathbf{x}_{(N)}\right) r_x p_x^2 \mathbf{G}_{x^t x}\mathbf{G}_{xx^s}p_{x^s}\mathbf{K}_{sv}y^v \\
&+ 2\alpha \sum_{tuv} y^u \mathbf{K}_{ut}p_{x^t}\int dx P\left(x|\mathbf{x}_{(N)}\right) r_x p_x \mathbf{G}_{x^t x}\mathbf{J}_{xx^v}y^v \\
&+ \sum_{uv} y^u \int dx P\left(x|\mathbf{x}_{(N)}\right) r_x \mathbf{J}_{x^u x}\mathbf{J}_{xx^v}y^v.
\end{aligned} \tag{13}$$

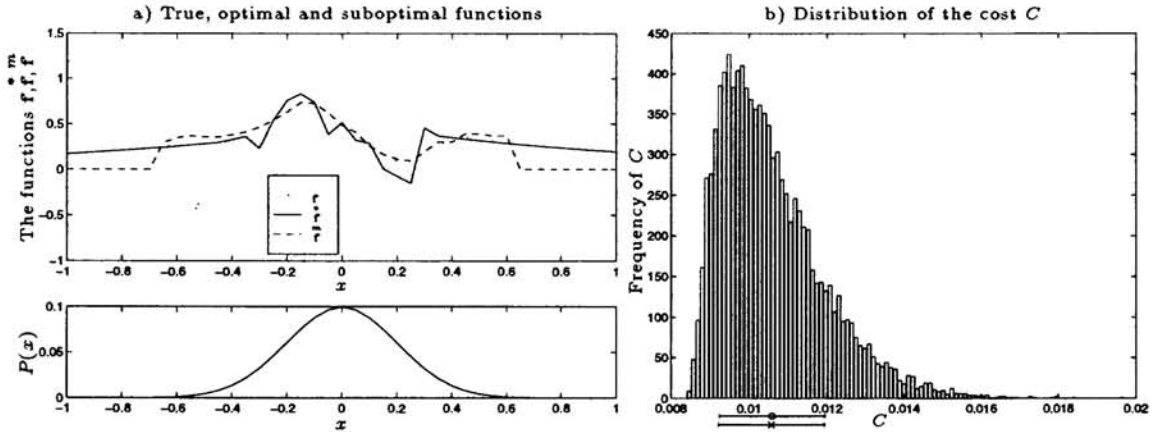

Figure 1: **a)** The lower figure shows the input distribution. The upper figure shows the true function $\mathbf{f}$ generated from a Gaussian prior with covariance matrix $\mathbf{V}$ (dotted line), the optimal function $\overset{*}{\mathbf{f}} = \mathbf{Ab}$ (solid line) and the suboptimal solution $\overset{m}{\mathbf{f}}$ (dashed line). **b)**The distribution of the cost function obtained by generating functions from the posterior Gaussian with covariance matrix $\mathbf{A}$ and calculating the cost according to equation 14. The mean and one standard deviation calculated analytically and numerically are shown by the lower and upper error bars respectively.

Taking $P\left(x|\mathbf{x}_{(N)}\right)$ to be Gaussian (the maximum likelihood estimate would be reasonable) and $p$, $r$, and $q$ uniform, the first four integrals are straightforward. The latter two involve the model $\mathbf{J}$, and were evaluated numerically in the work reported below.

## 5  Numerical results

We present one numerical example to illustrate the formalism, and another to illustrate its application.

For the first illustration, let the input and output variables be one dimensional real numbers. Let the input distribution $P(x)$ be a Gaussian with mean $\mu_x = 0$ and standard deviation $\sigma_x = 0.2$. Nearly all inputs then fall within the range $[-1, 1]$, which we uniformly quantise into 41 bins. The true function $\mathbf{f}$ is generated from a Gaussian distribution with $\boldsymbol{\mu}_f = 0$ and $41 \times 41$ covariance matrix $\mathbf{V}$ with elements $\mathbf{V}_{xx'} = e^{-|x-x'|}$. 50 training inputs $x$ were generated from the input distribution and assigned corresponding outputs $y = \mathbf{f}_x + \epsilon$, where $\epsilon$ is Gaussian noise with zero mean and standard deviation $\sqrt{q_x/\beta} \equiv 0.01$. The cost weight $r_x \equiv 1$.

The inputs were thermometer coded[3] over 256 bits, from which 100 subsets of 30 bits were randomly selected. Each of the $100 \times 2^{30}$ patterns formed over these bits defines a RAMnet feature which evaluates to 1 when that pattern is present in the input $x$. (Only those features which actually appear in the data need to be tabulated.) The 50 training data points were used in this way to train an n-tuple

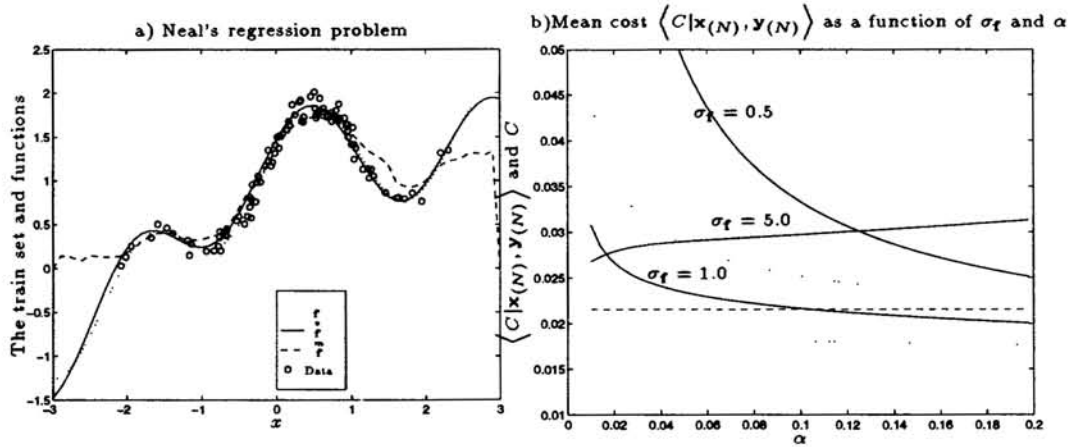

Figure 2: a) Neal's Regression problem. The true function $\mathbf{f}$ is indicated by a dotted line, the optimal function $\overset{*}{\mathbf{f}}$ is denoted by a solid line and the suboptimal solution $\overset{m}{\mathbf{f}}$ is indicated by a dashed line. Circles indicate the training data. b) Dependence of the cost prediction on the values of parameters $\alpha$ and $\sigma_\mathbf{f}$. The cost evaluated from the test set is plotted as a dashed line, predicted cost is shown as a solid line with one standard deviation indicated by a dotted line.

regression network. The input distribution and functions $\mathbf{f}$, $\overset{m}{\mathbf{f}}$, $\overset{*}{\mathbf{f}}$ are plotted in figure 1a.

A Gaussian distribution with mean $\overset{*}{\mathbf{f}}$ and posterior covariance matrix $\mathbf{A}$ was then used to generate $10^4$ functions. For each such function $\mathbf{fp}$, the generalisation cost

$$C = \tfrac{1}{2}\sum_x P\left(x|\mathbf{x}_{(N)}\right)(\mathbf{fp}_x - \overset{m}{\mathbf{f}}_x)^2. \tag{14}$$

was computed. A histogram of these costs appears in figure 1b, together with the theoretical and numerically computed average generalisation cost and its variance. Good agreement is evident.

Another one-dimensional problem illustrates the use of this formalism for predicting the generalisation performance of a RAMnet when the prior over functions can only be guessed. The true function, taken from (Neal, 1995) is given by

$$\mathbf{f}_x = 0.3 + 0.4x + 0.5\sin(2.7x) + 1.1/(1+x^2) + \epsilon \tag{15}$$

where the Gaussian noise variable $\epsilon$ has mean $\mu_\epsilon = 0$ and standard deviation $\sqrt{q_x/\beta} \equiv 0.1$. The cost weight $r_x \equiv 1$. The training and test set each comprised 100 data-points. The inputs were generated by the standard normal distribution ($\mu_x = 0$, $\sigma_x = 1$) and converted into the binary strings using a thermometer code. The input range $[-3, 3]$ was quantised into 61 uniform bins.

The training set and the functions $\mathbf{f}$, $\overset{m}{\mathbf{f}}$, $\overset{*}{\mathbf{f}}$ are shown on figure 2a for $\alpha = 0.1$. The function space covariance matrix was defined to have the Gaussian form $\mathbf{V}_{xx'} = e^{-\tfrac{1}{2}\frac{(x-x')^2}{\sigma_\mathbf{f}^2}}$ where $\sigma_\mathbf{f} = 1.0$.

$\sigma_f$ is the correlation length of the functions, which is of order 1, judging from figure 2a. The overall scale of variation is $1/\sqrt{\alpha}$, which appears to be about 3, so $\alpha$ should be about 1/9. Figure 2b shows the expected cost as a function of $\alpha$ for various choices of $\sigma_f$, with error bars on the $\sigma_f = 1.0$ curve. The actual cost computed from the test set according to $C = \frac{1}{2} \sum_i^m (y^i - \mathbf{f}_x)^2$ is plotted with a dashed line. There is good agreement around the sensible values of $\alpha$ and $\sigma_f$.

## 6 Conclusions

This paper demonstrates that unusual models, such as the ultra-fast RAMnets which are not trained by directly optimising a cost function, can be analysed in a Bayesian framework to determine their generalisation cost. Because the formalism is constructed in terms of distributions over function space rather than distributions over model parameters, it can be used for model comparison, and in particular to select RAMnet parameters.

The main drawback with this technique, as it stands, is the need to numerically integrate two expressions which involve the model. This difficulty intensifies rapidly as the input dimension increases. Therefore, it is now a research priority to search for RAMnet feature sets which allow these integrals to be performed analytically.

It would also be interesting to average the expected costs over the training data, producing an expected generalisation cost for an algorithm. The $\mathbf{y}_{(N)}$ integral is straightforward, but the $\mathbf{x}_{(N)}$ integral is difficult. However, similar integrals have been carried out in the thermodynamic limit (high input dimension) (Sollich, 1994), so the investigation of these techniques in the current setting is another promising research direction.

## 7 Acknowledgements

We would like to thank the Aston Neural Computing Group, and especially Huaiyu Zhu, Chris Williams, and David Saad for helpful discussions.

## Footnotes

[1]In physics, this is essentially the mode function expansion of $U^{-1}$, the differential operator with Green's function $U$.

[2]$V^{-1}$ needs to be a compact operator for this to work in the infinite-dimensional limit.

[3]The first $256(x+1)/2$ bits are set to 1, and the remaining bits to 0.

## References

Allinson, N.M., & Kołcz, A. 1995. *N-tuple Regression Network*. to be published in Neural Networks.

Neal, R. 1995. *Introductory documentation for software implementing Bayesian learning for neural networks using Markov chain Monte Carlo techniques*. Tech. rept. Dept of Computer Science, University of Toronto.

Rohwer, R., & Morciniec, M. 1996. A theoretical and experimental account of the n-tuple classifier performance. *Neural Computation*, **8**(3), 657–670.

Sollich, Peter. 1994. Finite-size effects in learning and generalization in linear perceptrons. *J. Phys. A*, **27**, 7771–7784.

Zhu, H., & Rohwer, R. to appear 1996. Bayesian regression filters and the issue of priors. *Neural Computing and Applications*. ftp://cs.aston.ac.uk/neural/zhuh/reg_fil_prior.ps.Z.